# On the connections between saliency and tracking

**Vijay Mahadevan**
Yahoo! Labs
Bangalore, India
vmahadev@yahoo-inc.com

**Nuno Vasconcelos**
Statistical Visual Computing Laboratory
UC San Diego, La Jolla, CA 92092
nuno@ece.ucsd.edu

## Abstract

A model connecting visual tracking and saliency has recently been proposed. This model is based on the *saliency hypothesis for tracking* which postulates that tracking is achieved by the top-down tuning, based on target features, of discriminant center-surround saliency mechanisms over time. In this work, we identify three main predictions that must hold if the hypothesis were true: 1) tracking reliability should be larger for salient than for non-salient targets, 2) tracking reliability should have a dependence on the defining variables of saliency, namely feature contrast and distractor heterogeneity, and must replicate the dependence of saliency on these variables, and 3) saliency and tracking can be implemented with common low level neural mechanisms. We confirm that the first two predictions hold by reporting results from a set of human behavior studies on the connection between saliency and tracking. We also show that the third prediction holds by constructing a common neurophysiologically plausible architecture that can computationally solve both saliency and tracking. This architecture is fully compliant with the standard physiological models of V1 and MT, and with what is known about attentional control in area LIP, while explaining the results of the human behavior experiments.

## 1  Introduction

Biological vision systems have evolved sophisticated tracking mechanisms, capable of tracking complex objects, undergoing complex motion, in challenging environments.These mechanisms have been an area of active research in both neurophysiology [10, 34] and psychophysics [28], where research has been devoted to the study of object tracking by humans [29]. This effort has produced several models of multi-object tracking, that account for the experimental evidence from human psychometric data [28]. Prominent among these are the FINST model of Pylyshyn [29], and the object file model of Kahnemann et al [18]. However, these models are not quantitative, and only explain the psychophysics of tracking simple stimuli, such as dots or bars. They do not specify a set of computations for the implementation of a general purpose tracking algorithm, and it is unclear how they could be applied to natural scenes. While some computational models for multiple object tracking (MOT) such as the oscillatory neural network model of Kazanovich et al. [19], and the particle filter based model of Vul et al. [37], have been proposed, there have been no attempts to demonstrate their applicability to real video scenes.

Visual tracking has also been widely studied in computer vision, where numerous tracking algorithms [38] have been proposed. Early solutions relied on simple object representations, and emphasized the prediction of object dynamics, typically using a Kalman filter. The prediction of these dynamics turned out to be difficult, motivating the introduction of more sophisticated methods such as particle filtering [15]. Nevertheless, because these approaches relied on simple target representations, they could not deal with complex scenes. This motivated research in the *appearance-based modeling* techniques [17, 32, 9] where a model of object appearance is learned from the target location in the initial frame, and used to identify the target in the next. It is, however, difficult to learn appearance models from complex scenes, where background detail tends to drift into the region used to learn the model, corrupting the learning.

The best results among tracking algorithms have recently been demonstrated for a class of methods that pose object tracking as incremental target/background classification [22, 8, 2, 13]. These *discriminant trackers* train a classifier to distinguish target from background at each frame. This classifier is then used to determine the location of the target in the next frame. Target and background are extracted at this location, the classifier updated, and the process iterated.

Recent work in the computer vision literature [22] has postulated a connection between discriminant tracking and one of the core processes of early biological vision - saliency, by suggesting that the ability to track objects is a side-effect of the saliency mechanisms that are known to guide the deployment of attention. More precisely, [22] has hypothesized that *tracking is a simple consequence of object-based tuning, over time, of the mechanisms used by the attentional system to implement bottom-up saliency*. We refer to this as the *saliency hypothesis for tracking*. Working under this hypothesis, [22] proposed a tracker based on the *discriminant saliency* principle of [12]. This is a principle for bottom-up center-surround saliency, which poses saliency as discrimination between a target (center) and a null (surround) hypothesis. Center-surround discriminant saliency has previously been shown to predict various psychophysical traits of human saliency and visual search performance [11]. The extension proposed by [22], to the tracking problem, endows discriminant saliency with a top-down feature selection mechanism. This mechanism enhances features that respond strongly to the target and weakly to the background, transforming the saliency operation from a search for locations where center is distinct from the surround, to a search for locations where target is present in the center but not in the surround. [22] has shown that this tracker has state-of-the-art performance on a number of tracking benchmarks from the computer vision literature.

In this work, we evaluate the validity of the saliency hypothesis by identifying three main predictions that ensue from the saliency hypothesis: 1) tracking reliability should be larger for salient than for non-salient targets, 2) tracking reliability should have a dependence on the defining variables of saliency, namely feature contrast and distractor heterogeneity, and must replicate the dependence of saliency on these variables and, 3) saliency and tracking can be implemented with common low level neural mechanisms. We confirm that the first two of these predictions hold by performing several human behavior experiments on the dependence between target saliency and human tracking performance. These experiments build on well understood properties of saliency, such as pop-out effects, to show that tracking requires discrimination between target and background using a center-surround mechanism. In addition, we characterize the dependence of tracking performance on the extent of discrimination, by gradually varying feature contrast between target and distractors in the tracking tasks. The results show that both tracking performance and saliency have highly similar patterns of dependency on feature contrast and distractor heterogeneity. To confirm that the third prediction holds, we show that both saliency and tracking can be implemented by a network compliant with the widely accepted neurophysiological models of neurons in area V1 [5] and the middle temporal area (MT) [36], and with the emerging view of attentional control in the lateral intra-parietal area (LIP) [3]. This network extends the substantial connections between discriminant saliency and the standard model that have already been shown [12] and is a biologically plausible optimal model for *both* saliency and tracking.

## 2   Human Behavior Experiments on Saliency and Tracking

We start by reporting on human behavior experiments[1] investigating the connections between the psychophysics of tracking and saliency. To the best of our knowledge, this is the first report on psychophysics experiments studying the relation between attentional tracking of a single target and its saliency. Video stimuli were designed with the Psychtoolbox [4] on Matlab v7, running on a Windows XP PC. A 27 inch LCD monitor of size $47.5° \times 30°$ visual angle and resolution of $1270 \times 1068$ pixels was used to present the stimuli. Subjects were at a viewing distance of 57 cm. The same apparatus was used for all experiments.

### 2.1   Experiment 1 : Saliency affects tracking performance

The experimental setting was inspired by the tracking paradigm of Pylyshyn [29]. Subjects viewed displays containing a *green* target disk surrounded by 70 *red* distractor disks and a static fixation square. Example displays are shown in [1]. At the start of each trial, the target disk was cued with a bounding box. Subjects were instructed to track the target covertly, while their eyes fixated on a black fixation square in the center. After a keystroke from the subject, all disks moved independently, with random motion, for around 7 seconds. Then, the disks stopped moving and the colors of 3

disks were switched to 3 new colors - cyan, magenta and blue. Of these, one was the target and the other two the spatially closest distractors. Subjects were asked to identify the target among the 3 highlighted disks.

**Method** 13 subjects (age 22-35, 9 male, 4 female) performed 4 trials each, organized into 2 versions of 2 conditions. First version: this version tested subject tracking performance under two different stimulus conditions. In the first, denoted *salient*, the target remained green throughout the presentation, changing randomly to one of the three highlight colors at the end of the 7 seconds. In the second, denoted *non-salient*, the target remained green for the first half of this period, switched to red for the remaining time, finally turning to a highlight color. While in the first condition the target is salient throughout the presentation, the second makes the target non-salient throughout the latter half of the trial. To eliminate potential effects of any other variables (e.g. target-distractor distances and motion patterns), the non-salient display was created by rotating each frame of a salient display by $90°$ (and changing the green disk to red in the second half of the presentation).

Under the saliency hypothesis for tracking, the rate of successful target tracking should be much higher for salient than for non-salient displays. However, this could be due to the fact that the target was the only green disk in salient displays, and since it continuously popped-out subjects could be acquiring the target at any time even after losing track. Second Version: The second version ruled out this alternate hypothesis by using a different type of display for the *salient* condition. In this case, the target was a red disk, and its 7 nearest spatial neighbors were green. All other distractors were randomly assigned to either the red or green class. This eliminated the percept of pop-out. As before, the display for the non-salient condition was created by rotation and color switch of the target on the second half of the presentation. The video displays are available in the attached supplement [1].

**Results and Discussion** Figure 2 (a) and (b) present the rate of successful tracking in the two versions. In both cases, this rate was much higher in the *salient* than in the *non-salient* condition. In the latter, the tracking performance was almost at the chance level of $\frac{1}{3}$, suggesting complete tracking failure. In fact, the similarity of detection rates in the two experiments suggests that target pop-out does not aid human tracking performance at all. It only matters if the target is *locally* salient, i.e. salient with respect to its immediate neighborhood. This is consistent with the saliency hypothesis, since bottom-up saliency mechanisms are well known to have a center-surround structure [16, 12]. In fact, it suggests two new predictions. The first, motivated by the hypothesis that tracking requires top-down biases of bottom-up saliency, is that center-surround organization also applies to tracking. To address this prediction, we will investigate the spatial organization of tracking mechanisms in greater detail in Experiment 3. The second, which follows from the fact that only target color varied between the two conditions, is that tracking performance depends on the *discriminability* of the target. We study this prediction in Experiment 2. While the first experiment used color as a discriminant cue, the conclusion that saliency affect tracking performance applies even when other features are salient. For example, studies on multiple object tracking with identical targets and distractors have reported tracking failure when target and distractors are too close to each other [14]. This is consistent with the discriminant hypothesis: when target and distractors are identical, the target must be spatio-temporally salient (due to its trajectory or position) in an immediate neighborhood to be tracked accurately.

## 2.2 Experiment 2: Tracking reliability as a function of feature contrast

Experiment 2 aimed to investigate the connection between the two phenomena in greater detail, namely to *quantify* how tracking reliability depends on target saliency. Since saliency is not an independent variable, this quantification can only be done indirectly. One possibility is to resort to a variable commonly used to manipulate saliency: the amount of *feature contrast* between target and distractors. Several features can be used, as it is well known that targets which differ from distractors in terms of color, luminance, orientation or texture can be perceived as salient [27, 25]. In fact, Nothdurft [26] has precisely *quantified* the dependence of saliency on orientation contrast in static displays. His work has shown that perceived target saliency increases with the orientation contrast between target and neighboring distractors. This increase is quite non-linear, exhibiting the threshold and saturation effects shown in Figure 1 (a), where we present curves of saliency as a function of orientation contrast between target and distractors for three levels of distractor homogeneity [26]. The relationship between tracking reliability and target saliency can thus be characterized by manipulating orientation contrast and measuring the impact on tracking performance. If the saliency

hypothesis for tracking holds, saliency and tracking reliability should be equivalent functions of orientation contrast. In particular, increasing orientation contrast between target and distractors should result in a non-linear increase of tracking reliability, with threshold and saturation effects similar to those observed by Nothdurft.

**Method** 12 subjects (8 male and 4 female) in the age range 21-35 participated in the study. The experimental setting was adapted from the work of Makovski and Jiang [23]. All displays had size $26° \times 26°$ ($700 \times 700$ pixels) and consisted of 23 ellipses, all of color blue, against a black background. Each ellipse had a major axis of $\sim 0.56°$ (15 pixels) and minor axis of $\sim 0.19°$ (5 pixels). The orientation of the ellipses depended on the condition from which the trial was drawn. At the start of a trial, one of the ellipses was designated as target (cued with a white bounding box). Subjects were instructed to track the target covertly, while fixating on a white square at the center of the screen. On a keystroke, the ellipses started moving and continued to do so for $\sim 8$-10 sec. At the end of the trial, all ellipses were completely occluded by larger white disks and subjects asked to click on the disk corresponding to the target. Each subject performed 30 trials under 7 conditions, for a total of 210 trials, and no feedback was given on the accuracy of their selection.

The 7 conditions corresponded to different levels of orientation contrast between target and distractor ellipses. Distractor orientation, defined by the major axis of the distractor ellipses, was always $0°$. Target orientation, determined by the major axis of the target ellipse, was selected from 7 values: $0°$, $10°$, $20°$, $30°$, $40°$, $60°$ or $80°$. This made orientation contrast equal to the target orientation. Example displays are shown in the attached supplement [1]. To keep all other variables (e.g. distance between items, motion patterns, distance from target to fixation square) identical, a trial was first created for one condition (target orientation $0°$). The trials of all other conditions were obtained by applying a transformation to each frame of this video clip. This consisted of an affine transformation of the grid of ellipse centers, followed by the desired change in target orientation.

To study the effect of distractor heterogeneity [26], three versions of the experiment were conducted with different numbers of ellipses in the target orientation. In the first version, only one ellipse (the actual target) was in target orientation. In this case, there was no distractor heterogeneity. In the second version, 18 of the 23 ellipses were in distractor orientation, and the remaining 5 in target orientation. One of the latter was the actual target. Finally, in the third version, 13 ellipses were in distractor and 10 in target orientation, for the largest degree of distractor heterogeneity.

**Results and Discussion** Figure 1 (b), shows the curves of tracking accuracy vs. orientation contrast obtained in all three versions of the experiment. These curves are remarkably similar to Nothdurft's saliency curves, shown in (a). Again, there are 1) distinct threshold and saturation effects for tracking, with tracking accuracy saturating for orientation contrasts beyond $40°$, and 2) decreasing tracking accuracy as distractor heterogeneity increases. The co-variation of tracking accuracy and saliency is illustrated in Figure 1 (c), where the two quantities are presented as a scatter plot The correlation between the two variable is near perfect ($r = 0.975$). In summary, tracking has a dependence on orientation contrast remarkably similar to that of saliency.

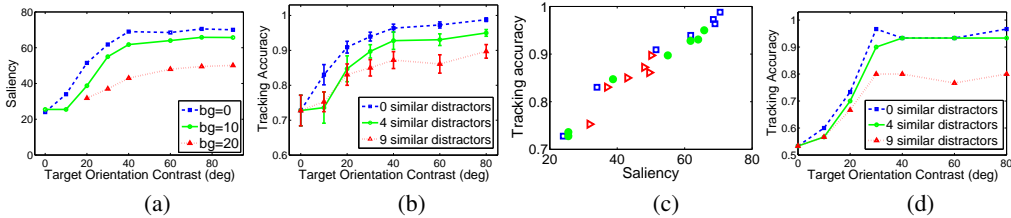

Figure 1: (a) saliency vs. orientation contrast (adapted from [26]) (b) human tracking success rate vs. orientation contrast. (c) scatter plot of saliency values from (a) vs tracking accuracy from (b), $r = 0.975$. (d) model prediction : tracking success rate vs. orientation contrast for the network of Figure 3.

## 2.3 Experiment 3: The spatial structure of tracking

It is well known that bottom-up saliency mechanisms are based on spatially localized center-surround processing [16, 6]. Hence, the saliency hypothesis for tracking predicts that tracking performance depends only on distractors within a spatial neighborhood of the target. The results

of Experiment 2 provide some evidence in support of this prediction, by showing that tracking performance depends on distractor heterogeneity. This implies that the visual content of the background affects human tracking performance. The open question is whether the effect of the background 1) is limited to a *localized* neighborhood of the target, or 2) extends to the entire field of view. This question motivated Experiment 3. In this experiment, the distance $d_{csd}$ between the target and the closest distractor of the same orientation, denoted the *closest similar distractor* (CSD), was controlled so that $d_{csd} = R$, where $R$ is a parameter. This guaranteed that there were no distractors with the target orientation inside a neighborhood of radius $R$ around it. By jointly varying this parameter and the amount of distractor heterogeneity, it is possible to test three hypotheses: (a) no surround region is involved in tracking: in this case, the rate of tracking success does not depend on the distractor heterogeneity at all, (b) the entire visual field affects tracking performance: in this case, for a fixed distractor heterogeneity, the rate of tracking success does not depend on $R$, (c) the effect of the surround is spatially localized: in this case, there is a critical radius $R_{critical}$ beyond which distractors have no influence in tracking performance. This implies that the rate of tracking success does not depend on distractor heterogeneity for $R > R_{critical}$. Experiment 2 already established that hypothesis (a) does not hold. Experiment 3 was designed to determine which of (b) and (c) holds.

**Method**    9 subjects (7 male and 2 female) in the age range 21-35 participated in the study. The target orientation was fixed at $40°$ for all stimuli. Two versions of the experiment were conducted, with two levels of distractor heterogeneity. As in Experiment 2, the first version used 18 (5) of the 23 ellipses in distractor (target) orientation. In the second version, 13 ellipses were in distractor and 10 in target orientation. In both versions, the stimulus was produced with four values of average $R$ (average, over all frames in the video sequence, of the distance $d_{csd}$). Across the 4 conditions, this quantity was in the range $1.67°$ to $5.01°$ (about 45 pixels to 135 pixels).

**Results and Discussion**    Figure 2(a) presents the rate of tracking success as a function of average $R$, for the two versions of the experiment. The tracking accuracy for the case where there is no distractor heterogeneity (no distractor with the target orientation) is also shown, as a flat line. Two main observations are worth noting. First, for a fixed (non-zero) amount of distractor heterogeneity, tracking performance always increases with $R$. This implies that it is easier to track when the CSD is farther from the target. Second, for large $R$ tracking accuracy does not depend on distractor heterogeneity (it is nearly the same under the two heterogeneity conditions), converging to the accuracy observed when there is no distractor heterogeneity (Experiment 3). These observations support the conclusion that hypothesis (c) holds, i.e. tracking ability is influenced by a *localized* surround region, of radius $R_{critical} \approx 4°$. When similar distractors are kept out of this region, the degree of distractor heterogeneity has no effect in tracking performance.

In summary, results of the human behavior experiments show that the first two predictions made by the saliency hypothesis for tracking hold. These predictions are that tracking reliability 1) is larger for salient than for non-salient targets (Experiment 1), 2) depends on the defining variables of saliency, namely feature contrast and distractor heterogeneity (Experiment 2), and replicates the dependence of saliency on these variables. This includes the threshold and saturation effects of the dependence of saliency on feature contrast (Experiment 2), and the spatially localized dependence of saliency on distractor heterogeneity (Experiment 3). Overall, these experiments provide strong evidence in support of the saliency hypothesis for tracking. We next consider the final prediction, which is that saliency and tracking can be implemented with common neural mechanisms.

## 3   Joint neural architecture for saliency and tracking

To construct a saliency based neurally plausible computational model for tracking we start with the neural model proposed by [12] to compute saliency and identify the mechanisms required to extend it to perform tracking, and then show how these mechanisms can be implemented in a biologically plausible manner.

In [12], saliency is equated to optimal decision-making between two classes of visual stimuli, with label $C \in \{0, 1\}$, $C = 1$ for stimuli in a *target* class, and $C = 0$ for stimuli in a *background* class. The classes are defined in a center-surround manner where, at each location $l$, the target (background) class is that of stimuli in a center (surround) window. The stimuli are not observed directly, but through projection onto a set of $n$ features, of responses $\mathbf{Y}(l) = (Y_1(l), \ldots, Y_n(l))$. The saliency of location $l$ is then equated to the expected accuracy of target/background classification,

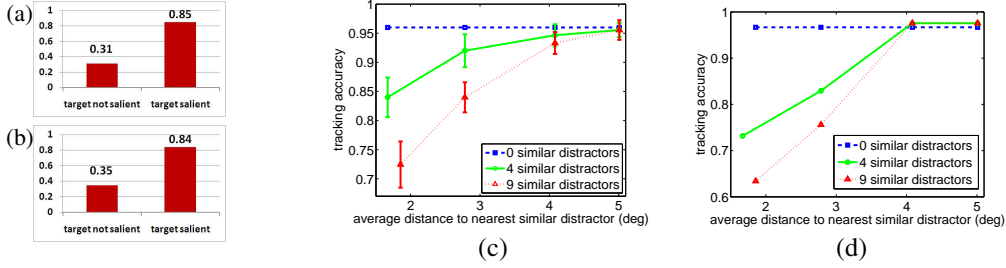

Figure 2: (a) and (b) Experiment 1: successful target tracking rate for targets that are (a) globally salient (pop-out), and (b) locally salient (do not pop-out). (c) and (d) Experiment 3: the effect of background on tracking performance - (c) Tracking accuracy of human subjects for two versions of distractor homogeneities are plotted as a function of the average target-similar distractor distance. Also shown, using a blue dashed line, is the tracking accuracy for the version with no similar distractors at target orientation of $40°$ from Experiment 2. (d) model prediction for the same data using the saliency based model of Figure 3.

given the feature responses from the two classes and can be written as:

$$S(l) = \frac{1}{n}\sum_k S_k(l), \qquad S_k(l) = E_{Y(l)}\{\gamma[P_{C(l)|Y_k(l)}(1|y)]\}, \qquad \gamma(x) = \left\{ \begin{array}{ll} x - \frac{1}{2} & x \geq 0.5 \\ 0 & x < 0.5 \end{array} \right. \qquad (1)$$

The saliency measure $S_k(l)$ is the expected confidence with which the feature response $Y_k(l)$ is assigned to the target class. $\gamma(x)$ is a nonlinearity that thresholds the posterior probability $P_{C(l)|Y_k(l)}(1|y)$ to prevent locations assigned to the background class by the Bayes decision rule $(P_{C(l)|Y_k(l)}(1|y) \leq 0.5)$, from contributing to the saliency. This tunes the saliency measure to respond only to the presence of target stimuli, not to its absence. This definition of saliency was shown, in [12], to be computable using units that conform to the standard neurophysiological model of cells in visual cortex area V1 [5], when the features are bandpass filters (e.g. Gabor filters) extracted from static natural images. However, for the tracking task, the feature set $\mathbf{Y}$ for representing the target and background needs to contain spatiotemporal features that are tuned to the velocity of moving patterns. It can be shown that saliency for such velocity tuned spatiotemporal features can be computed by combining the outputs of a set of V1 like units of [12], akin to the widely used approach for constructing models for MT cells from afferent V1 units [36, 33]. This enhanced network, illustrated in Figure 3, is equivalent to an MT neuron tuned to a particular velocity (see supplement [1]).

## 3.1 Neurophysiologically plausible feature selection

A key component of the saliency tracker of [22] is a feature selection procedure that continuously adapts the saliency measure of (1) to the target. The basic idea is to select, at each time step, the features in $\mathbf{Y}(l)$ that best discriminate between target (center) and background. This changes the saliency from a bottom-up identification of locations where center and surround differ, to a top-down identification of locations containing the target in the center and background in the surround. However, the procedure of [22] (based on feature ranking) is not biologically plausible. To derive a biologically plausible feature selection mechanism, we replace the saliency measure of (1) with a feature-weighted extension

$$S(l) = \sum_k \alpha_k S_k(l), \qquad \sum_k \alpha_k = 1 \qquad (2)$$

where $\alpha_k$ is the weight given to the saliency of the $k^{th}$ feature channel. We associate a binary variable $F_k$ with each feature $Y_k$, such that $F_k = 1$ if and only if $Y_k$ is the *most salient* feature of the target. We then assume that, given the knowledge of which feature is most salient, target presence at location $l$ is independent of the remaining feature responses, and so the posterior probability of target presence given the observation of all features is written as:

$$P_{C(l)|\mathbf{Y}(l),F_k}(1|\mathbf{y},1) = 2\gamma[P_{C(l)|Y_k(l)}(1|y)], \qquad (3)$$

This reflects a conservative strategy, where features cannot be considered salient unless they are individually discriminant for target presence. Given the location $l^*$ where the target has been detected, the posterior probability of feature saliency can then be computed by Bayes rule

$$P_{F_k|C(l^*)}(1|1) = \frac{P_{C(l^*)|F_k}(1|1)P_{F_k}(1)}{\sum_j P_{C(l^*)|F_j}(1|1)P_{F_j}(1)}, \quad \text{where} \qquad (4)$$

$$P_{C(l^*)|F_k}(1|1) = \int P_{C(l^*)|\mathbf{Y}(l^*),F_k}(1|\mathbf{y},1)P_{\mathbf{Y}(l^*)|F_k}(\mathbf{y}|1)d\mathbf{y} \tag{5}$$

$$= \int 2\gamma[P_{C(l^*)|Y_k(l^*)}(1|y)]P_{Y_k(l^*)}(y)dy \quad \text{(using (3))}$$

$$= 2E_{Y_k(l^*)}\{\gamma[P_{C(l^*)|Y_k(l^*)}(1|y)]\} = 2S_k(l^*), \tag{6}$$

and the last equality follows from (1). Using (6) in (4), we get

$$P_{F_k|C(l^*)}(1|1) = \frac{S_k(l^*)P_{F_k}(1)}{\sum_j S_j(l^*)P_{F_j}(1)}. \tag{7}$$

Under reasonable assumptions of persistence of the dominant features in the target, this analysis can be extended over time, by denoting the state of $F_k$ and $l^*$ at time $t$ by $F_k^t$ and $l_t^*$, and the sequence of target locations till time $t$ by $\mathbf{l}_t^* = (l_t^*, l_{t-\tau}^* \ldots l_0^*)$, and we get the recursion (see [1]),

$$P_{F_k^t|C(\mathbf{l}_t^*)}(1|\mathbf{1}) = \frac{S_k(l_t^*)P_{F_k^{t-\tau}|C(\mathbf{l}_{t-\tau}^*)}(1|\mathbf{1})}{\sum_j S_j(l_t^*)P_{F_j^{t-\tau}|C(\mathbf{l}_{t-\tau}^*)}(1|\mathbf{1})}. \tag{8}$$

Hence, the posterior probability of feature $k$ being the most salient at time $t$ given that the target is at $l_t^*$ is computed by divisively normalizing a weighted version of $S_k(l_t^*)$, the bottom-up saliency of the feature k at $l_t^*$, by the total saliency summed over all features. The weight applied to the saliency of each feature (corresponding to $\alpha_k$ in (2)) is the posterior probability of the feature being the most salient at time $t - \tau$. Therefore the posterior at time $t - \tau$ is fed back with a delay, to become the weight at time $t$. This *enhances the most salient features, suppressing the non-salient ones,* and is equivalent to applying a soft-thresholding to select only the dominant features.

This feature selection mechanism involving selective enhancement and suppression of features, operating on the outputs of the MT stage bears a close resemblance to the phenomenon of feature-based attention [24]. In fact, the proposed approach to feature selection is similar to previously proposed biologically plausible models of feature-based attention that rely on a Bayesian formulation and include divisive normalization [30, 31, 20, 7]. Further, neurophysiological studies have found evidence of feature-based attention in the lateral intraparietal (LIP) area [3]. LIP is also known to have cortico-cortical connections to area MT [21], and attentional control is thought to be fed-back from LIP to MT [35]. Studies also suggest that the LIP might be computing a priority map that combines both bottom-up inputs and top-down signals, and the peak of this map response is used to guide visual attention [3]. These findings are compatible with the feature selection approach of (8), and therefore area LIP is a plausible candidate location for the feature selection stage of our model.

### 3.2 Neurophysiologically plausible discriminant tracker

A neurophysiologically plausible version of the discriminant tracker of [22] can be constructed with the discriminant saliency measure of (1), and the feature selection mechanism of (8). As in [22], in the absence of top-level information regarding the target, initialization and target acquisition can be treated as discrimination between the visual stimulus contained in a pair of *center* (target) and *surround* windows, at every location of the visual field. In this case, there is no explicit top-down guidance about the object to recognize, and the saliency of location $l$ is measured by the saliency of *all* unmodulated feature responses. This consists of using the bottom-up saliency measure of (2) with $\alpha_k = P_{F_k^0}(1)$, where $P_{F_k^0}(1)$ is a uniform prior for feature selection, at time $t = 0$. The outputs of all features or neurons are then summed with equal weights to produce a final saliency map. The peak of this map represents the location which is most distinct from its surround, based on the responses of the motion sensitive spatio-temporal features, and becomes the target. Spatial attention is then shifted to the peak of this map.

Once the initial target location is attended, the feature selection mechanism modulates the saliency response of the individual feature channels, using the weights of (8). The final saliency value at that location also becomes the normalizing constant for the divisive normalization of (8). These feature weights are fed back to MT neurons, where each feature map is enhanced or attenuated depending on the corresponding feature weight. This *enhances the features that are salient for target detection, and suppresses the non-salient ones.* LIP also feeds back the retinotopic weight map corresponding to spatial attention, causing a suppression of feature responses in all areas other than in a neighborhood of the current locus of attention.

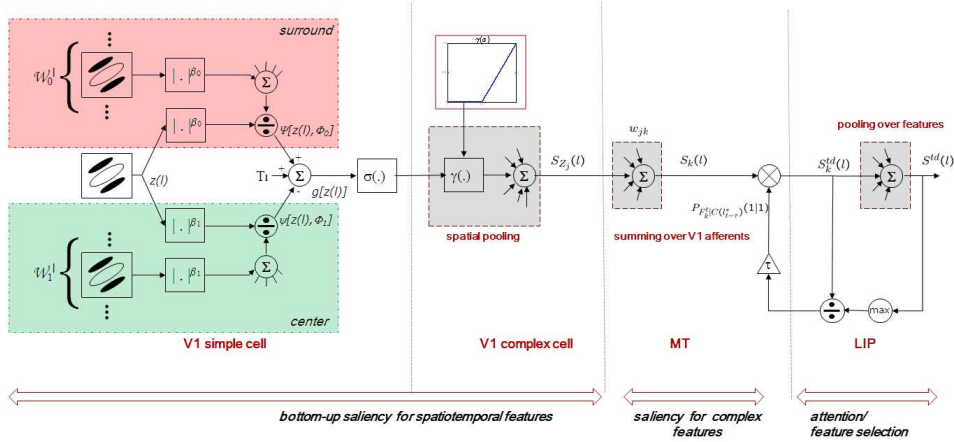

Figure 3: The network for tracking using feature selection. The discriminant saliency network of [12] is used to construct a model for an MT neuron. Feature selection, performed possibly in area LIP with weights being fed-back to MT, is achieved by the modulation of the response of each feature channel by its saliency value after divisive normalization across features.

After the latency due to feedback, say at time $t + \tau$, the new feature weights and spatial weights, modulate the feature maps, which are again fed forward to LIP, where the updated saliency map is computed by simple summation. The top-down saliency of location $l$ at time $t + \tau$ is then given by

$$S^{td}(l) = \sum_j S_j^{td}(l) = \sum_j S_j(l) P_{F_j^t|C(\mathbf{l}_t^*)}(1|\mathbf{1}). \tag{9}$$

where $S_j(l)$ is the modulated saliency response of the $j^{th}$ feature.

Spatial attention suppresses all but a neighborhood of the last known target location $l_t^*$, and the feature-based attention suppresses all features except those present in the target and discriminative with respect to the background. Therefore, the peak of the new saliency map corresponds to the position that best resembles the target at time $t + \tau$, and attention is shifted to that position.

$$l_{t+\tau}^* = \operatorname*{argmax}_l S^{td}(l) \tag{10}$$

The process is iterated, so as to track the target over time, as in [22]. The entire tracking network is shown in Figure 3. The computation, in V1, of $S_{Z_j}(l)$ is implemented with the bottom-up network of [12]. V1 outputs are then linearly combined with weights $w_{jk}$ (which are described in supplement [1]) to obtain the MT responses $S_k(l)$. The remaining operations, possibly in LIP, compute the probabilities of (8) and the top-down saliency map of (9).

## 4  Validation of joint architecture

We applied the network of Figure 3 to the sequences used in Experiment 1. Representative frames of the result of tracking on the displays of the experiment and the videos are available from [1]. The model replicates the trend observed in both versions of the experiment, accurately tracking the target in the salient conditions, and losing track in the non-salient condition.

The results of applying the network to the stimuli in Experiments 2 and 3 are shown in Figures 1(d) and 2(d) respectively. It is seen that the model predictions accurately match the trend observed in all three versions of the Experiment 2. The model also predicts the effect of background seen in Experiment 3.

## 5  Conclusion

We provide the first *verifiable evidence* for a connection between saliency and tracking that was earlier only hypothesized [22]. In particular, we show that three main predictions of the hypothesis hold. First, using psychophysics experiments we show that tracking requires discrimination between target and background using a center-surround mechanism, and that tracking reliability and saliency have a common dependence on feature contrast and distractor heterogeneity. Next we construct a tracking model starting a neurally plausible architecture to compute saliency, and show that it can be implemented with widely accepted models of cortical computation. Specifically, the model is based on a feature selection mechanism akin to the well known phenomenon of feature-based attention in MT. Finally, we show that the tracking model accurately replicates all our psychophysics results.

## Footnotes

[1]IRB approved study, subjects provided informed consent and were compensated $8 per hour

# References

[1] See attached supplementary material.

[2] S. Avidan. Ensemble tracking. *IEEE PAMI*, 29(2):261–271, 2007.

[3] J. Bisley & M. Goldberg, "Attention, intention, & priority in the parietal lobe," *Annu. Rev. Neurosci*, 33, p. 1–21, 2010.

[4] D. H. Brainard. The psychophysics toolbox. *Spatial Vision*, 10:433–436, 1997.

[5] M. Carandini et al., Do we know what the early visual system does? *J. Neuroscience*, 25, 2005.

[6] J. Cavanaugh, W. Bair, & J. Movshon. Nature & interaction of signals from the receptive field center and surround in macaque V1 neurons. *J. Neurophysiol.*, 88:2530–2546, 2002.

[7] S. Chikkerur, et al., What & where: A Bayesian inference theory of attention. *Vision Research*, 2010.

[8] R. Collins, Y. Liu, & M. Leordeanu. On-line selection of discriminative tracking features. *IEEE PAMI*, 27(10):1631 – 1643, October 2005.

[9] D. Comaniciu, V. Ramesh, & P. Meer. Kernel-based object tracking. *IEEE PAMI*, 25(5):564–577, 2003.

[10] J. C. Culham, et al., Cortical fmri activation produced by attentive tracking of moving targets. *J. Neurophysiol*, 80(5):2657–2670, 1998.

[11] D. Gao, V. Mahadevan, & N. Vasconcelos. On the plausibility of the discriminant center-surround hypothesis for visual saliency. *Journal of Vision*, 8(7):1–18, 6 2008.

[12] D. Gao & N. Vasconcelos. Decision-theoretic saliency: computational principle, biological plausibility, & implications for neurophysiology & psychophysics. *Neural Computation*, 21:239–271, Jan 2009.

[13] H. Grabner & H. Bischof. On-line boosting & vision. *IEEE CVPR*, 1:260–267, 2006.

[14] J. Intriligator & P. Cavanagh. The spatial resolution of visual attention. *Cog. Psych.*, 43:171–216, 1997.

[15] M. Isard & A. Blake. Condensation: conditional density propagation for visual tracking. *IJCV*, 29, 1998.

[16] L. Itti et al., A model of saliency-based visual attention for rapid scene analysis. *IEEE PAMI*, 20(11):1254–1259, 1998.

[17] A. D. Jepson et al., Robust online appearance models for visual tracking. *IEEE PAMI*, 25(10), 2003.

[18] D. Kahneman, A. Treisman, & B. J. Gibbs. The reviewing of object files: Object-specific integration of information. *Cognitive Psychology*, 24(2):175–219, 1992.

[19] Y. Kazanovich & R. Borisyuk. An oscillatory neural model of multiple object tracking. *Neural computation*, 18(6):1413–1440, 2006.

[20] J. Lee & J. Maunsell. A normalization model of attentional modulation of single unit responses. *PLoS One*, 4(2), 2009.

[21] J. Lewis & D. Van Essen, "Corticocortical connections of visual, sensorimotor, & multimodal processing areas in the parietal lobe of the macaque monkey," *J. Comparative Neurol.*, 428(1), p. 112–137, 2000.

[22] V. Mahadevan & N. Vasconcelos. Saliency-based discriminant tracking. CVPR, 2009.

[23] T. Makovski & Y. Jiang. Feature binding in attentive tracking of distinct objects. *Visual cognition*, 17(1):180–194, 2009.

[24] J. Maunsell & S. Treue. Feature-based attention in visual cortex. *Trends in Neurosci.*, 29(6), 2006.

[25] H. C. Nothdurft. Texture segmentation & pop-out from orientation contrast. *Vision Research*, 31(6):1073–1078, 1991.

[26] H. C. Nothdurft. The conspicuousness of orientation & motion contrast. *Spatial Vision*, 7:341–363, 1993.

[27] H. C. Nothdurft. Salience from feature contrast: additivity across dimensions. *Vision Research*, 40:1183–1201, 2000.

[28] L. Oksama & J. Hyn. Is multiple object tracking carried out automatically by an early vision mechanism independent of higher-order cognition? *Visual Cognition*, 11(5):631 – 671, 2004.

[29] Z. W. Pylyshyn & R. W. Storm. Tracking multiple independent targets: evidence for a parallel tracking mechanism. *Spatial vision*, 3(3):179–197, 1988.

[30] R. Rao. Bayesian inference & attentional modulation in the visual cortex. *Neuroreport*, 16(16), 2005.

[31] J. Reynolds & D. Heeger. The normalization model of attention. *Neuron*, 61(2):168–185, 2009.

[32] D. Ross et al., Incremental learning for robust visual tracking. *IJCV*, 77(1-3):125–141, 2008.

[33] N. Rust et al., How MT cells analyze the motion of visual patterns. *Nat. Neurosci.*, 9(11), 2006.

[34] H. Sakata, H. Shibutani, & K. Kawano. Functional properties of visual tracking neurons in posterior parietal association cortex of the monkey. *J Neurophysiol*, 49(6):1364–1380, 1983.

[35] Y. Saalmann, I. Pigarev, & T. Vidyasagar, "Neural mechanisms of visual attention: how top-down feedback highlights relevant locations," *Science*, 316(5831), p. 1612, 2007.

[36] E. Simoncelli & D. Heeger. A model of neuronal responses in visual area MT. *Vision Research*, 38(5):743–761, 1998.

[37] E. Vul et al., Explaining human multiple object tracking as resource-constrained approximate inference in a dynamic probabilistic model. *NIPS*, 22:1955–1963, 2009.

[38] A. Yilmaz, O. Javed, & M. Shah. Object tracking: A survey. *ACM Computing Surveys*, 38(4):13, 2006.

